# Graph Laplacian Regularization for Large-Scale Semidefinite Programming

**Kilian Q. Weinberger**
Dept of Computer and Information Science
U of Pennsylvania, Philadelphia, PA 19104
`kilianw@seas.upenn.edu`

**Fei Sha**
Computer Science Division
UC Berkeley, CA 94720
`feisha@cs.berkeley.edu`

**Qihui Zhu**
Dept of Computer and Information Science
U of Pennsylvania, Philadelphia, PA 19104
`qihuizhu@seas.upenn.edu`

**Lawrence K. Saul**
Dept of Computer Science and Engineering
UC San Diego, La Jolla, CA 92093
`saul@cs.ucsd.edu`

## Abstract

In many areas of science and engineering, the problem arises how to discover low
dimensional representations of high dimensional data. Recently, a number of re-
searchers have converged on common solutions to this problem using methods
from convex optimization. In particular, many results have been obtained by con-
structing semidefinite programs (SDPs) with low rank solutions. While the rank
of matrix variables in SDPs cannot be directly constrained, it has been observed
that low rank solutions emerge naturally by computing high variance or maximal
trace solutions that respect local distance constraints. In this paper, we show how
to solve very large problems of this type by a matrix factorization that leads to
much smaller SDPs than those previously studied. The matrix factorization is de-
rived by expanding the solution of the original problem in terms of the bottom
eigenvectors of a graph Laplacian. The smaller SDPs obtained from this matrix
factorization yield very good approximations to solutions of the original problem.
Moreover, these approximations can be further refined by conjugate gradient de-
scent. We illustrate the approach on localization in large scale sensor networks,
where optimizations involving tens of thousands of nodes can be solved in just a
few minutes.

## 1   Introduction

In many areas of science and engineering, the problem arises how to discover low dimensional repre-
sentations of high dimensional data. Typically, this high dimensional data is represented in the form
of large graphs or matrices. Such data arises in many applications, including manifold learning [12],
robot navigation [3], protein clustering [6], and sensor localization [1]. In all these applications, the
challenge is to compute low dimensional representations that are consistent with observed measure-
ments of local proximity. For example, in robot path mapping, the robot's locations must be inferred
from the high dimensional description of its state in terms of sensorimotor input. In this setting,
we expect similar state descriptions to map to similar locations. Likewise, in sensor networks, the
locations of individual nodes must be inferred from the estimated distances between nearby sensors.
Again, the challenge is to find a planar representation of the sensors that preserves local distances.

In general, it is possible to formulate these problems as simple optimizations over the low dimen-
sional representations $\vec{x}_i$ of individual instances (e.g., robot states, sensor nodes). The most straight-

forward formulations, however, lead to non-convex optimizations that are plagued by local minima. For this reason, large-scale problems cannot be reliably solved in this manner.

A more promising approach reformulates these problems as convex optimizations, whose global minima can be efficiently computed. Convexity is obtained by recasting the problems as optimizations over the inner product matrices $\mathbf{X}_{ij} = \vec{x}_i \cdot \vec{x}_j$. The required optimizations can then be relaxed as instances of semidefinite programming [10], or SDPs. Two difficulties arise, however, from this approach. First, only low rank solutions for the inner product matrices $\mathbf{X}$ yield low dimensional representations for the vectors $\vec{x}_i$. Rank constraints, however, are non-convex; thus SDPs and other convex relaxations are not guaranteed to yield the desired low dimensional solutions. Second, the resulting SDPs do not scale very well to large problems. Despite the theoretical guarantees that follow from convexity, it remains prohibitively expensive to solve SDPs over matrices with (say) tens of thousands of rows and similarly large numbers of constraints.

For the first problem of "rank regularization", an apparent solution has emerged from recent work in manifold learning [12] and nonlinear dimensionality reduction [14]. This work has shown that while the rank of solutions from SDPs cannot be directly constrained, low rank solutions often emerge naturally by computing maximal trace solutions that respect local distance constraints. Maximizing the trace of the inner product matrix $\mathbf{X}$ has the effect of maximizing the variance of the low dimensional representation $\{\vec{x}_i\}$. This idea was originally introduced as "semidefinite embedding" [12, 14], then later described as "maximum variance unfolding" [9] (and yet later as "kernel regularization" [6, 7]). Here, we adopt the name maximum variance unfolding (MVU) which seems to be currently accepted [13, 15] as best capturing the underlying intuition.

This paper addresses the second problem mentioned above: how to solve *very large* problems in MVU. We show how to solve such problems by approximately factorizing the large $n \times n$ matrix $\mathbf{X}$ as $\mathbf{X} \approx \mathbf{Q}\mathbf{Y}\mathbf{Q}^\top$ where $\mathbf{Q}$ is a pre-computed $n \times m$ rectangular matrix with $m \ll n$. The factorization leaves only the much smaller $m \times m$ matrix $\mathbf{Y}$ to be optimized with respect to local distance constraints. With this factorization, and by collecting constraints using the Schur complement lemma, we show how to rewrite the original optimization over the large matrix $\mathbf{X}$ as a simple SDP involving the smaller matrix $\mathbf{Y}$. This SDP can be solved very quickly, yielding an accurate approximation to the solution of the original problem. Moreover, if desirable, this solution can be further refined [1] by (non-convex) conjugate gradient descent in the vectors $\{\vec{x}_i\}$.

The main contribution of this paper is the matrix factorization that makes it possible to solve large problems in MVU. Where does the factorization come from? Either implicitly or explicitly, all problems of this sort specify a graph whose nodes represent the vectors $\{\vec{x}_i\}$ and whose edges represent local distance constraints. The matrix factorization is obtained by expanding the low dimensional representation of these nodes (e.g., sensor locations) in terms of the $m \ll n$ bottom (smoothest) eigenvectors of the graph Laplacian. Due to the local distance constraints, one expects the low dimensional representation of these nodes to vary smoothly as one traverses edges in the graph. The presumption of smoothness justifies the partial orthogonal expansion in terms of the bottom eigenvectors of the graph Laplacian [5]. Similar ideas have been widely applied in graph-based approaches to semi-supervised learning [4]. Matrix factorizations of this type have also been previously studied for manifold learning; in [11, 15], though, the local distance constraints were not properly formulated to permit the large-scale applications considered here, while in [8], the approximation was not considered in conjunction with a variance-maximizing term to favor low dimensional representations.

The approach in this paper applies generally to any setting in which low dimensional representations are derived from an SDP that maximizes variance subject to local distance constraints. For concreteness, we illustrate the approach on the problem of localization in large scale sensor networks, as recently described by [1]. Here, we are able to solve optimizations involving tens of thousands of nodes in just a few minutes. Similar applications to the SDPs that arise in manifold learning [12], robot path mapping [3], and protein clustering [6, 7] present no conceptual difficulty.

This paper is organized as follows. Section 2 reviews the problem of localization in large scale sensor networks and its formulation by [1] as an SDP that maximizes variance subject to local distance constraints. Section 3 shows how we solve large problems of this form—by approximating the inner product matrix of sensor locations as the product of smaller matrices, by solving the smaller SDP that results from this approximation, and by refining the solution from this smaller SDP using

local search. Section 4 presents our experimental results on several simulated networks. Finally, section 5 concludes by discussing further opportunities for research.

## 2   Sensor localization via maximum variance unfolding

The problem of sensor localization is best illustrated by example; see Fig. 1. Imagine that sensors are located in major cities throughout the continental US, and that nearby sensors can estimate their distances to one another (e.g., via radio transmitters). From only this local information, the problem of sensor localization is to compute the individual sensor locations and to identify the whole network topology. In purely mathematical terms, the problem can be viewed as computing a low rank embedding in two or three dimensional Euclidean space subject to local distance constraints.

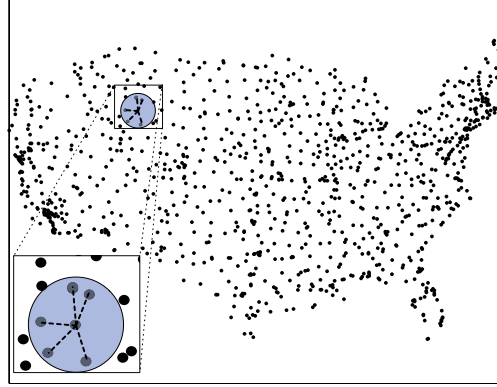

We assume there are $n$ sensors distributed in the plane and formulate the problem as an optimization over their planar coordinates $\vec{x}_1, \ldots, \vec{x}_n \in \Re^2$. (Sensor localization in three dimensional space can be solved in a similar way.) We define a neighbor relation $i \sim j$ if the $i$th and $j$th sensors are sufficiently close to

Figure 1: Sensors distributed over US cities. Distances are estimated between nearby cities within a fixed radius.

estimate their pairwise distance via limited-range radio transmission. From such (noisy) estimates of local pairwise distances $\{d_{ij}\}$, the problem of sensor localization is to infer the planar coordinates $\{\vec{x}_i\}$. Work on this problem has typically focused on minimizing the sum-of-squares loss function [1] that penalizes large deviations from the estimated distances:

$$\min_{\vec{x}_1, \ldots, \vec{x}_n} \sum_{i \sim j} \left( \|\vec{x}_i - \vec{x}_j\|^2 - d_{ij}^2 \right)^2 \tag{1}$$

In some applications, the locations of a few sensors are also known in advance. For simplicity, in this work we consider the scenario where no such "anchor points" are available as prior knowledge, and the goal is simply to position the sensors up to a global rotation, reflection, and translation. Thus, to the above optimization, without loss of generality we can add the centering constraint:

$$\left\| \sum_i \vec{x}_i \right\|^2 = 0. \tag{2}$$

It is straightforward to extend our approach to incorporate anchor points, which generally leads to even better solutions. In this case, the centering constraint is not needed.

The optimization in eq. (1) is not convex; hence, it is likely to be trapped by local minima. By relaxing the constraint that the sensor locations $\vec{x}_i$ lie in the $\Re^2$ plane, we obtain a convex optimization that is much more tractable [1]. This is done by rewriting the optimization in eqs. (1–2) in terms of the elements of the inner product matrix $\mathbf{X}_{ij} = \vec{x}_i \cdot \vec{x}_j$. In this way, we obtain:

$$
\begin{aligned}
&\textbf{Minimize:} \quad && \sum_{i \sim j} \left( \mathbf{X}_{ii} - 2\mathbf{X}_{ij} + \mathbf{X}_{jj} - d_{ij}^2 \right)^2 \\
&\textbf{subject to:} \quad && \textbf{(i) } \sum_{ij} \mathbf{X}_{ij} = 0 \quad \textbf{and} \quad \textbf{(ii) } \mathbf{X} \succeq 0.
\end{aligned}
\tag{3}
$$

The first constraint centers the sensors on the origin, as in eq. (2), while the second constraint specifies that $\mathbf{X}$ is positive semidefinite, which is necessary to interpret it as an inner product matrix in Euclidean space. In this case, the vectors $\{\vec{x}_i\}$ are determined (up to rotation) by singular value decomposition.

The convex relaxation of the optimization in eqs. (1–2) drops the constraint that that the vectors $\vec{x}_i$ lie in the $\Re^2$ plane. Instead, the vectors will more generally lie in a subspace of dimensionality

equal to the rank of the solution $\mathbf{X}$. To obtain planar coordinates, one can project these vectors into their two dimensional subspace of maximum variance, obtained from the top two eigenvectors of $\mathbf{X}$. Unfortunately, if the rank of $\mathbf{X}$ is high, this projection loses information. As the error of the projection grows with the rank of $\mathbf{X}$, we would like to enforce that $\mathbf{X}$ has low rank. However, the rank of a matrix is not a convex function of its elements; thus it cannot be directly constrained as part of a convex optimization.

Mindful of this problem, the approach to sensor localization in [1] borrows an idea from recent work in unsupervised learning [12, 14]. Very simply, an extra term is added to the loss function that favors solutions with high variance, or equivalently, solutions with high trace. (The trace is proportional to the variance assuming that the sensors are centered on the origin, since $\mathbf{tr}(X) = \sum_i \|\vec{x}_i\|^2$.) The extra variance term in the loss function favors low rank solutions; intuitively, it is based on the observation that a flat piece of paper has greater diameter than a crumpled one. Following this intuition, we consider the following optimization:

$$
\begin{aligned}
&\textbf{Maximize:} \quad \mathbf{tr}(\mathbf{X}) - \nu \sum\nolimits_{i \sim j} \left( \mathbf{X}_{ii} - 2\mathbf{X}_{ij} + \mathbf{X}_{jj} - d_{ij}^2 \right)^2 \\
&\textbf{subject to:} \quad \text{(i) } \sum\nolimits_{ij} \mathbf{X}_{ij} = 0 \quad \textbf{and} \quad \text{(ii) } \mathbf{X} \succeq 0.
\end{aligned}
\tag{4}
$$

The parameter $\nu > 0$ balances the trade-off between maximizing variance and preserving local distances. This general framework for trading off global variance versus local rigidity has come to be known as *maximum variance unfolding* (MVU) [9, 15, 13].

As demonstrated in [1, 9, 6, 14], these types of optimizations can be written as semidefinite programs (SDPs) [10]. Many general-purpose solvers for SDPs exist in the public domain (e.g., [2]), but even for systems with sparse constraints, they do not scale very well to large problems. Thus, for small networks, this approach to sensor localization is viable, but for large networks ($n \sim 10^4$), exact solutions are prohibitively expensive. This leads us to consider the methods in the next section.

## 3 Large-scale maximum variance unfolding

Most SDP solvers are based on interior-point methods whose time-complexity scales cubically in the matrix size and number of constraints [2]. To solve large problems in MVU, even approximately, we must therefore reduce them to SDPs over small matrices with small numbers of constraints.

### 3.1 Matrix factorization

To obtain an optimization involving smaller matrices, we appeal to ideas in spectral graph theory [5]. The sensor network defines a connected graph whose edges represent local pairwise connectivity. Whenever two nodes share an edge in this graph, we expect the locations of these nodes to be relatively similar. We can view the location of the sensors as a function that is defined over the nodes of this graph. Because the edges represent local distance constraints, we expect this function to vary smoothly as we traverse edges in the graph. The idea of graph regularization in this context is best understood by analogy. If a smooth function is defined on a bounded interval of $\Re^1$, then from real analysis, we know that it can be well approximated by a low order Fourier series. A similar type of low order approximation exists if a smooth function is defined over the nodes of a graph. This low-order approximation on graphs will enable us to simplify the SDPs for MVU, just as low-order Fourier expansions have been used to regularize many problems in statistical estimation.

Function approximations on graphs are most naturally derived from the eigenvectors of the graph Laplacian [5]. For unweighted graphs, the graph Laplacian $\mathbf{L}$ computes the quadratic form

$$
f^\top \mathbf{L} f = \sum_{i \sim j} (f_i - f_j)^2
\tag{5}
$$

on functions $f \in \Re^n$ defined over the nodes of the graph. The eigenvectors of $\mathbf{L}$ provide a set of basis functions over the nodes of the graph, ordered by smoothness. Thus, smooth functions $f$ can be well approximated by linear combinations of the bottom eigenvectors of $\mathbf{L}$.

Expanding the sensor locations $\vec{x}_i$ in terms of these eigenvectors yields a compact factorization for the inner product matrix $\mathbf{X}$. Suppose that $\vec{x}_i \approx \sum_{\alpha=1}^{m} \mathbf{Q}_{i\alpha} \vec{y}_\alpha$, where the columns of the $n \times m$

rectangular matrix $\mathbf{Q}$ store the $m$ bottom eigenvectors of the graph Laplacian (excluding the uniform eigenvector with zero eigenvalue). Note that in this approximation, the matrix $\mathbf{Q}$ can be cheaply pre-computed from the unweighted connectivity graph of the sensor network, while the vectors $\vec{y}_\alpha$ play the role of unknowns that depend in a complicated way on the local distance estimates $d_{ij}$. Let $\mathbf{Y}$ denote the $m \times m$ inner product matrix of these vectors, with elements $\mathbf{Y}_{\alpha\beta} = \vec{y}_\alpha \cdot \vec{y}_\beta$. From the low-order approximation to the sensor locations, we obtain the matrix factorization:

$$\mathbf{X} \approx \mathbf{QYQ}^\top. \tag{6}$$

Eq. (6) approximates the inner product matrix $\mathbf{X}$ as the product of much smaller matrices. Using this approximation for localization in large scale networks, we can solve an optimization for the much smaller $m \times m$ matrix $\mathbf{Y}$, as opposed to the original $n \times n$ matrix $\mathbf{X}$.

The optimization for the matrix $\mathbf{Y}$ is obtained by substituting eq. (6) wherever the matrix $\mathbf{X}$ appears in eq. (4). Some simplifications occur due to the structure of the matrix $\mathbf{Q}$. Because the columns of $\mathbf{Q}$ store mutually orthogonal eigenvectors, it follows that $\mathrm{tr}(\mathbf{QYQ}^\top) = \mathrm{tr}(\mathbf{Y})$. Because we do not include the uniform eigenvector in $\mathbf{Q}$, it follows that $\mathbf{QYQ}^\top$ automatically satisfies the centering constraint, which can therefore be dropped. Finally, it is sufficient to constrain $\mathbf{Y} \succeq 0$, which implies that $\mathbf{QYQ}^\top \succeq 0$. With these simplifications, we obtain the following optimization:

$$
\boxed{
\begin{array}{ll}
\textbf{Maximize:} & \mathrm{tr}(\mathbf{Y}) - \nu \sum_{i \sim j} \left[ (\mathbf{QYQ}^\top)_{ii} - 2(\mathbf{QYQ}^\top)_{ij} + (\mathbf{QYQ}^\top)_{jj} - d_{ij}^2 \right]^2 \\
\textbf{subject to:} & \mathbf{Y} \succeq 0
\end{array}
}
\tag{7}
$$

Eq. (6) can alternately be viewed as a form of regularization, as it constrains neighboring sensors to have nearby locations even when the estimated local distances $d_{ij}$ suggest otherwise (e.g., due to noise). Similar forms of graph regularization have been widely used in semi-supervised learning [4].

## 3.2   Formulation as SDP

As noted earlier, our strategy for solving large problems in MVU depends on casting the required optimizations as SDPs over small matrices with few constraints. The matrix factorization in eq. (6) leads to an optimization over the $m \times m$ matrix $\mathbf{Y}$, as opposed to the $n \times n$ matrix $\mathbf{X}$. In this section, we show how to cast this optimization as a correspondingly small SDP. This requires us to reformulate the quadratic optimization over $\mathbf{Y} \succeq 0$ in eq. (4) in terms of a linear objective function with linear or positive semidefinite constraints.

We start by noting that the objective function in eq. (7) is a quadratic function of the elements of the matrix $\mathbf{Y}$. Let $\mathcal{Y} \in \Re^{m^2}$ denote the vector obtained by concatenating all the columns of $\mathbf{Y}$. With this notation, the objective function (up to an additive constant) takes the form

$$b^\top \mathcal{Y} - \mathcal{Y}^\top \mathbf{A} \mathcal{Y}, \tag{8}$$

where $\mathbf{A} \in \Re^{m^2 \times m^2}$ is the positive semidefinite matrix that collects all the quadratic coefficients in the objective function and $b \in \Re^{m^2}$ is the vector that collects all the linear coefficients. Note that the trace term in the objective function, $\mathrm{tr}(\mathbf{Y})$, is absorbed by the vector $b$.

With the above notation, we can write the optimization in eq. (7) as an SDP in standard form. As in [8], this is done in two steps. First, we introduce a dummy variable $\ell$ that serves as a lower bound on the quadratic piece of the objective function in eq. (8). Next, we express this bound as a linear matrix inequality via the Schur complement lemma. Combining these steps, we obtain the SDP:

$$
\boxed{
\begin{array}{ll}
\textbf{Maximize:} & b^\top \mathcal{Y} - \ell \\[2mm]
\textbf{subject to:} & \textbf{(i) } \mathbf{Y} \succeq 0 \quad \textbf{and} \quad \textbf{(ii) } \begin{bmatrix} \mathbf{I} & \mathbf{A}^{\frac{1}{2}}\mathcal{Y} \\ (\mathbf{A}^{\frac{1}{2}}\mathcal{Y})^\top & \ell \end{bmatrix} \succeq 0.
\end{array}
}
\tag{9}
$$

In the second constraint of this SDP, we have used $\mathbf{I}$ to denote the $m^2 \times m^2$ identity matrix and $\mathbf{A}^{\frac{1}{2}}$ to denote the matrix square root. Thus, via the Schur lemma, this constraint expresses the lower bound $\ell \geq \mathcal{Y}^\top \mathbf{A} \mathcal{Y}$, and the SDP is seen to be equivalent to the optimization in eqs. (7–8).

The SDP in eq. (9) represents a drastic reduction in complexity from the optimization in eq. (7). The only variables of the SDP are the $m(m+1)/2$ elements of $\mathbf{Y}$ and the unknown scalar $\ell$. The

only constraints are the positive semidefinite constraint on $\mathbf{Y}$ and the linear matrix inequality of size $m^2 \times m^2$. *Note that the complexity of this SDP does not depend on the number of nodes or edges in the network.* As a result, this approach scales very well to large problems in sensor localization.

In the above formulation, it is worth noting the important role played by *quadratic* penalties. The use of the Schur lemma in eq. (9) was conditioned on the quadratic form of the objective function in eq. (7). Previous work on MVU has enforced the distance constraints as strict equalities [12], as one-sided inequalities [9, 11], and as soft constraints with linear penalties [14]. Expressed as SDPs, these earlier formulations of MVU involved as many constraints as edges in the underlying graph, even with the matrix factorization in eq. (6). Thus, the speed-ups obtained here over previous approaches are not merely due to graph regularization, but more precisely to its use in conjunction with quadratic penalties, all of which can be collected in a single linear matrix inequality via the Schur lemma.

### 3.3 Gradient-based improvement

While the matrix factorization in eq. (6) leads to much more tractable optimizations, it only provides an approximation to the global minimum of the original loss function in eq. (1). As suggested in [1], we can refine the approximation from eq. (9) by using it as initial starting point for gradient descent in eq. (1). In general, gradient descent on non-convex functions can converge to undesirable local minima. In this setting, however, the solution of the SDP in eq. (9) provides a highly accurate initialization. Though no theoretical guarantees can be made, in practice we have observed that this initialization often lies in the basin of attraction of the true global minimum.

Our most robust results were obtained by a two-step process. First, starting from the $m$-dimensional solution of eq. (9), we used conjugate gradient methods to maximize the objective function in eq. (4). Though this objective function is written in terms of the inner product matrix $\mathbf{X}$, the hill-climbing in this step was performed in terms of the vectors $\vec{x}_i \in \Re^m$. While not always necessary, this first step was mainly helpful for localization in sensor networks with irregular (and particularly non-convex) boundaries. It seems generally difficult to representation such boundaries in terms of the bottom eigenvectors of the graph Laplacian. Next, we projected the results of this first step into the $\Re^2$ plane and use conjugate gradient methods to minimize the loss function in eq. (1). This second step helps to correct patches of the network where either the graph regularization leads to oversmoothing and/or the rank constraint is not well modeled by MVU.

## 4 Results

We evaluated our algorithm on two simulated sensor networks of different size and topology. We did not assume any prior knowledge of sensor locations (e.g., from anchor points). We added white noise to each local distance measurement with a standard deviation of $10\%$ of the true local distance.

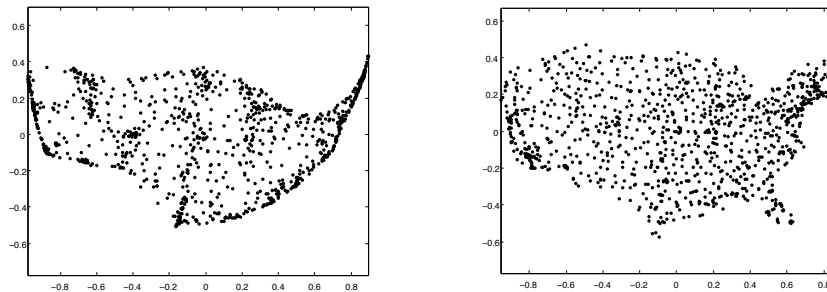

Figure 2: Sensor locations inferred for $n = 1055$ largest cities in the continental US. On average, each sensor estimated local distances to $18$ neighbors, with measurements corrupted by 10% Gaussian noise; see text. *Left:* sensor locations obtained by solving the SDP in eq. (9) using the $m = 10$ bottom eigenvectors of the graph Laplacian (computation time $4s$). Despite the obvious distortion, the solution provides a good initial starting point for gradient-based improvement. *Right:* sensor locations after post-processing by conjugate gradient descent (additional computation time $3s$).

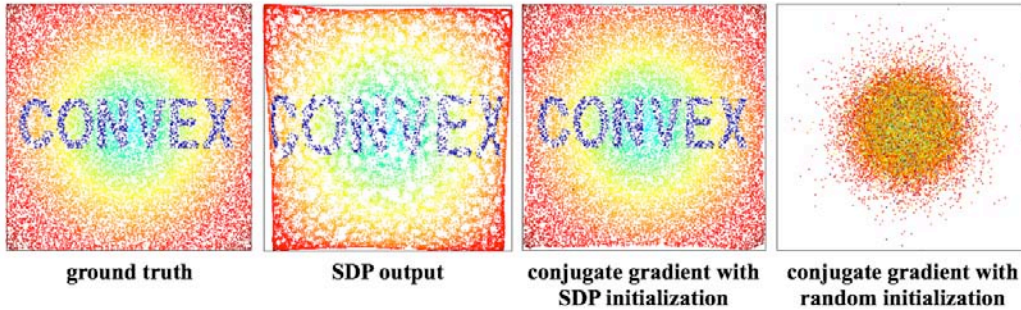

| ground truth | SDP output | conjugate gradient with SDP initialization | conjugate gradient with random initialization |

Figure 3: Results on a simulated network with $n = 20000$ uniformly distributed nodes inside a centered unit square. See text for details.

The first simulated network, shown in Fig. 1, placed nodes at scaled locations of the $n = 1055$ largest cities in the continental US. Each node estimated the local distance to up to $18$ other nodes within a radius of size $r = 0.09$. The SDP in eq. (9) was solved using the $m = 10$ bottom eigenvectors of the graph Laplacian. Fig. 2 shows the solution from this SDP (on the left), as well as the final result after gradient-based improvement (on the right), as described in section 3.3. From the figure, it can be seen that the solution of the SDP recovers the general topology of the network but tends to clump nodes together, especially near the boundaries. After gradient-based improvement, however, the inferred locations differ very little from the true locations. The construction and solution of the SDP required $4s$ of total computation time on a 2.4 GHz Pentium 4 desktop computer, while the post-processing by conjugate gradient descent took an additional $3s$.

The second simulated network, shown in Fig. 3, placed nodes at $n = 20000$ uniformly sampled points inside the unit square. The nodes were then centered on the origin. Each node estimated the local distance to up to 20 other nodes within a radius of size $r = 0.06$. The SDP in eq. (9) was solved using the $m = 10$ bottom eigenvectors of the graph Laplacian. The computation time to construct and solve the SDP was $19s$. The follow-up conjugate gradient optimization required $52s$ for 100 line searches. Fig. 3 illustrates the absolute positional errors of the sensor locations computed in three different ways: the solution from the SDP in eq. (8), the refined solution obtain by conjugate gradient descent, and the "baseline" solution obtained by conjugate gradient descent from a random initialization. For these plots, the sensors were colored so that the ground truth positioning reveals the word CONVEX in the foreground with a radial color gradient in the background.

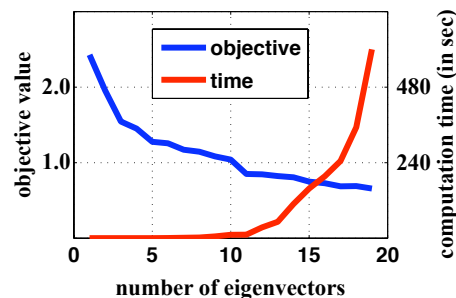

Figure 4: *Left:* the value of the loss function in eq. (1) from the solution of the SDP in eq. (8). *Right:* the computation time to solve the SDP. Both are plotted versus the number of eigenvectors, $m$, in the matrix factorization.

The refined solution in the third panel is seen to yield highly accurate results. (Note: the representations in the second and fourth panels were scaled by factors of 0.50 and 1028, respectively, to have the same size as the others.)

We also evaluated the effect of the number of eigenvectors, $m$, used in the SDP. (We focused on the role of $m$, noting that previous studies [1, 7] have thoroughly investigated the role of parameters such as the weight constant $\nu$, the sensor radius $r$, and the noise level.) For the simulated network with nodes at US cities, Fig. 4 plots the value of the loss function in eq. (1) obtained from the solution of eq. (8) as a function of $m$. It also plots the computation time required to create and solve the SDP. The figure shows that more eigenvectors lead to better solutions, but at the expense of increased computation time. In our experience, there is a "sweet spot" around $m \approx 10$ that best manages this tradeoff. Here, the SDP can typically be solved in seconds while still providing a sufficiently accurate initialization for rapid convergence of subsequent gradient-based methods.

Finally, though not reported here due to space constraints, we also tested our approach on various data sets in manifold learning from [12]. Our approach generally reduced previous computation times of minutes or hours to seconds with no noticeable loss of accuracy.

## 5  Discussion

In this paper, we have proposed an approach for solving large-scale problems in MVU. The approach makes use of a matrix factorization computed from the bottom eigenvectors of the graph Laplacian. The factorization yields accurate approximate solutions which can be further refined by local search. The power of the approach was illustrated by simulated results on sensor localization. The networks in section 4 have far more nodes *and* edges than could be analyzed by previously formulated SDPs for these types of problems [1, 3, 6, 14]. Beyond the problem of sensor localization, our approach applies quite generally to other settings where low dimensional representations are inferred from local distance constraints. Thus we are hopeful that the ideas in this paper will find further use in areas such as robotic path mapping [3], protein clustering [6, 7], and manifold learning [12].

**Acknowledgments**

This work was supported by NSF Award 0238323.

## References

[1]  P. Biswas, T.-C. Liang, K.-C. Toh, T.-C. Wang, and Y. Ye. Semidefinite programming approaches for sensor network localization with noisy distance measurements. *IEEE Transactions on Automation Science and Engineering*, 3(4):360–371, 2006.

[2]  B. Borchers. CSDP, a C library for semidefinite programming. *Optimization Methods and Software 11(1):613-623*, 1999.

[3]  M. Bowling, A. Ghodsi, and D. Wilkinson. Action respecting embedding. In *Proceedings of the Twenty Second International Conference on Machine Learning (ICML-05)*, pages 65–72, Bonn, Germany, 2005.

[4]  O. Chapelle, B. Schölkopf, and A. Zien, editors. *Semi-Supervised Learning*. MIT Press, Cambridge, MA, 2006.

[5]  F. R. K. Chung. *Spectral Graph Theory*. American Mathematical Society, 1997.

[6]  F. Lu, S. Keles, S. Wright, and G. Wahba. Framework for kernel regularization with application to protein clustering. *Proceedings of the National Academy of Sciences*, 102:12332–12337, 2005.

[7]  F. Lu, Y. Lin, and G. Wahba. Robust manifold unfolding with kernel regularization. Technical Report 1108, Department of Statistics, University of Wisconsin-Madison, 2005.

[8]  F. Sha and L. K. Saul. Analysis and extension of spectral methods for nonlinear dimensionality reduction. In *Proceedings of the Twenty Second International Conference on Machine Learning (ICML-05)*, pages 785–792, Bonn, Germany, 2005.

[9]  J. Sun, S. Boyd, L. Xiao, and P. Diaconis. The fastest mixing Markov process on a graph and a connection to a maximum variance unfolding problem. *SIAM Review*, 48(4):681–699, 2006.

[10]  L. Vandenberghe and S. P. Boyd. Semidefinite programming. *SIAM Review*, 38(1):49–95, March 1996.

[11]  K. Q. Weinberger, B. D. Packer, and L. K. Saul. Nonlinear dimensionality reduction by semidefinite programming and kernel matrix factorization. In Z. Ghahramani and R. Cowell, editors, *Proceedings of the Tenth International Workshop on Artificial Intelligence and Statistics (AISTATS-05)*, pages 381–388, Barbados, West Indies, 2005.

[12]  K. Q. Weinberger and L. K. Saul. Unsupervised learning of image manifolds by semidefinite programming. In *Proceedings of the IEEE Conference on Computer Vision and Pattern Recognition (CVPR-04)*, volume 2, pages 988–995, Washington D.C., 2004. Extended version in *International Journal of Computer Vision*, 70(1): 77-90, 2006.

[13]  K. Q. Weinberger and L. K. Saul. An introduction to nonlinear dimensionality reduction by maximum variance unfolding. In *Proceedings of the Twenty First National Conference on Artificial Intelligence (AAAI-06)*, Cambridge, MA, 2006.

[14]  K. Q. Weinberger, F. Sha, and L. K. Saul. Learning a kernel matrix for nonlinear dimensionality reduction. In *Proceedings of the Twenty First International Conference on Machine Learning (ICML-04)*, pages 839–846, Banff, Canada, 2004.

[15]  L. Xiao, J. Sun, and S. Boyd. A duality view of spectral methods for dimensionality reduction. In *Proceedings of the Twenty Third International Conference on Machine Learning (ICML-06)*, pages 1041–1048, Pittsburgh, PA, 2006.
